# Grouping and dimensionality reduction by locally linear embedding

**Marzia Polito**
Division of Physics, Mathematics and Astronomy
California Institute of Technology
Pasadena, CA, 91125
*polito@caltech.edu*

**Pietro Perona**
Division of Engeneering and Applied Mathematics
California Institute of Technology
Pasadena, CA, 91125
*perona@caltech.edu*

## Abstract

Locally Linear Embedding (LLE) is an elegant nonlinear dimensionality-reduction technique recently introduced by Roweis and Saul [2]. It fails when the data is divided into separate groups. We study a variant of LLE that can simultaneously group the data and calculate local embedding of each group. An estimate for the upper bound on the intrinsic dimension of the data set is obtained automatically.

## 1 Introduction

Consider a collection of $N$ data points $X_i \in \mathbb{R}^D$. Suppose that, while the dimension $D$ is large, we have independent information suggesting that the data are distributed on a manifold of dimension $d << D$. In many circumstances it is beneficial to calculate the coordinates $Y_i \in \mathbb{R}^d$ of the data on the lower-dimensional manifold, both because the shape of the manifold may yield some insight in the process that produced the data, and because it is cheaper to store and manipulate the data when it is embedded in fewer dimensions. How can we compute such coordinates?

Principal component analysis (PCA) is a classical technique which works well when the data lie close to a flat manifold [1]. Elegant methods for dealing with data that is distributed on curved manifolds have been recently proposed [3, 2]. We study one of them, Locally Linear Embedding (LLE) [2], by Roweis and Saul. While LLE is not designed to handle data that are disconnected, i.e. separated into groups, we show that a simple variation of the method will handle this situation correctly. Furthermore, both the number of groups and the upper bound on the intrinsic dimension of the data may be estimated automatically, rather than being given a-priori.

## 2 Locally linear embedding

The key insight inspiring LLE is that, while the data may not lie close to a *globally linear* manifold, it may be *approximately locally linear*, and in this case each point may be approximated as a linear combination of its nearest neighbors. The coefficients of this linear combination carries the vital information for constructing a lower-dimensional linear embedding.

More explicitly: consider a data set $\{X_i\}_{i=1...,N} \in \mathbb{R}^D$. The local linear structure can be easily encoded in a sparse $N$ by $N$ matrix $W$, proceeding as follows.

The first step is to choose a criterion to determine the neighbors of each point. Roweis and Saul chose an integer number $K$ and pick, for every point, the $K$ points nearest to it. For each point $X_i$ then, they determine the linear combination of its neighbors which best approximates the point itself. The coefficients of such linear combinations are computed by minimizing the quadratic cost function:

$$\epsilon(W) = \sum_i |X_i - \sum_{j=1}^{N} W_{ij} X_j|^2 \tag{1}$$

while enforcing the constraints $W_{ij} = 0$ if $X_j$ is not a neighbor of $X_i$, and $\sum_{j=1}^{N} W_{ij} = 1$ for every $i$; these constraints ensure that the approximation of $X_i \approx \hat{X}_i = \sum_{j=1}^{N} W_{ij} X_j$ lies in the affine subspace generated by the $K$ nearest neighbors of $X_i$, and that the solution $W$ is translation-invariant. This least square problem may be solved in closed form [2].

The next step consists of calculating a set $\{Y_i\}_{i=1,...,N}$ of points in $\mathbb{R}^d$, reproducing as faithfully as possible the local linear structure encoded in $W$. This is done minimizing a cost function

$$\Phi(Y) = \sum_{i=1}^{N} |Y_i - \sum_{j=1}^{N} W_{ij} Y_j|^2 \tag{2}$$

To ensure the uniqueness of the solution two constraint are imposed: translation invariance by placing the center of gravity of the data in the origin, i.e. $\sum_i Y_i = 0$, and normalized unit covariance of the $Y_i$'s, i.e. $\frac{1}{N} \sum_{i=1}^{N} Y_i \otimes Y_i = I$.

Roweis and Saul prove that $\Phi(Y) = tr(Y^T M Y)$, where $M$ is defined as

$$M = (I - W)^T (I - W).$$

The minimum of the function $\Phi(Y)$ for the $d$-th dimensional representation is then obtained with the following recipe. Given $d$, consider the $d + 1$ eigenvectors associated to the $d + 1$ smallest eigenvalues of the matrix $M$. Then discard the very first one. The rows of the matrix $Y$ whose columns are given by such $d$ eigenvectors give the desired solution. The first eigenvector is discarded because it is a vector composed of all ones, with 0 as eigenvalue. As we shall see, this is true when the data set is 'connected'.

### 2.1 Disjoint components

In LLE every data point has a set of $K$ neighbors. This allows us to partition of the whole data set $X$ into $K$-*connected components*, corresponding to the intuitive visual notion of different 'groups' in the data set.

We say that a partition $X = \cup_i U_i$ is *finer* than a partition $X = \cup_j V_j$ if every $U_i$ is contained in some $V_j$. The partition in $K$-*connected components* is the finest

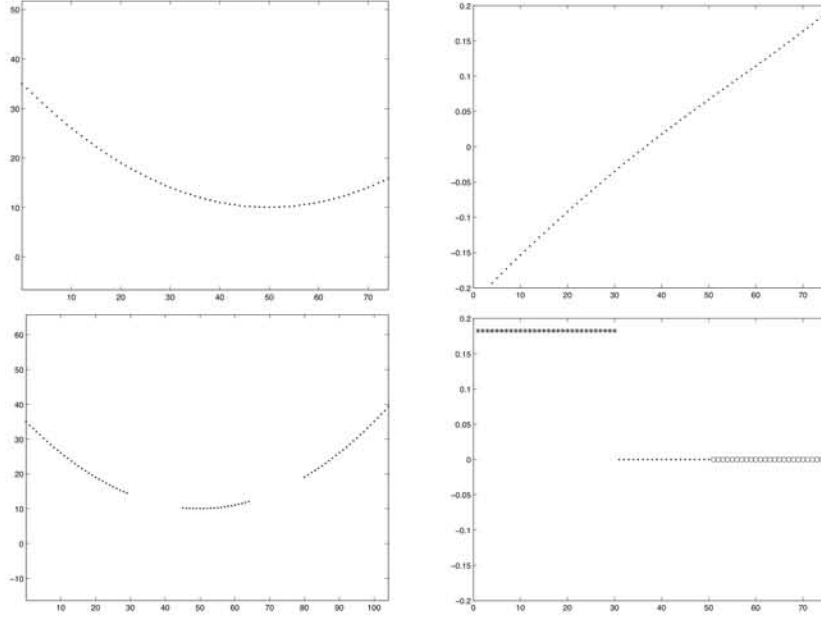

Figure 1: (Top-left) 2D data $X_i$ distributed along a curve (the index $i$ increases from left to right for convenience). (Top-right) Coordinates $Y_i$ of the same points calculated by LLE with $K = 10$ and $d = 1$. The $x$ axis represents the index $i$ and the $y$ axis represents $Y_i$. This is a good parametrization which recognizes the intrinsically 1-dimensional structure of the data. (Bottom-left) As above, the data is now disconnected, i.e. points in different groups do not share neighbors. (Bottom-right) One-dimensional LLE calculated on the data (different symbols used for points belonging to the different groups). Notice that the $Y_i$'s are not a good representation of the data any longer since they are constant within each group.

partition of the data set such that if two points have at least one neighbor in common, or one is a neighbor of the other, then they belong to the same component.

Note that for any two points in the same component, we can find an ordered sequence of points having them as endpoints, such that two consecutive points have at least one neighbor in common. A set is *K-connected* if it contains only one $K$-connected component.

Consider data that is not $K$-connected, then LLE does not compute a good parametrization, as illustrated in Figure 1.

## 2.2 Choice of $d$.

How is $d$ chosen? The LLE method [2] is based on the assumption that $d$ is known. What if we do not know it in advance? If we overestimate $d$ it then LLE behaves pathologically.

Let us consider a straight line, drawn in $\mathbb{R}^3$. Figure 2 shows what happens if $d$ is chosen equal to 1 and to 2. When the choice is 2 (right) then LLE 'makes up' information and generates a somewhat arbitrary 2D curve.

As an effect of the covariance constraint, the representation curves the line, the

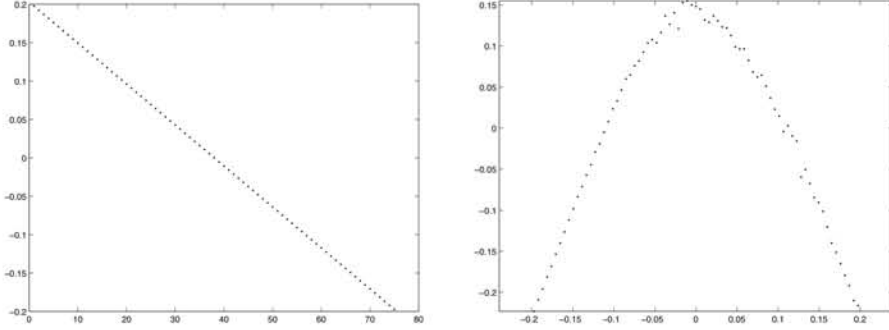

Figure 2: Coordinates $Y_i$ calculated for data $X_i$ distributed along a straight line in $\mathbb{R}^D = \mathbb{R}^3$ when the dimension $d$ is chosen as $d = 1$ (Left), and $d = 2$ (Right). The index $i$ is indicated along the $x$ axis (Left) and along the 2D curve (Right).

curvature can be very high, and even locally we possibly completely lose the linear structure. The problem is, we chosed the wrong target dimension. The one-dimensional LLE works in fact perfectly (see Figure 2, left).

PCA provides a principled way of estimating the intrinsic dimensionality of the data: it corresponds to the number of large singular values of the covariance matrix of the data. Is such an estimate possible with LLE as well?

## 3 Dimensionality detection: the size of the eigenvalues

In the example of Figure 2 the two dimensional representation of the data ($d = 2$) is clearly the 'wrong' one, since the data lie in a one-dimensional linear subspace. In this case the unit covariance constraint in minimizing the function $\Phi(Y)$ is not compatible with the linear structure. How could one have obtained the correct estimate of $d$? The answer is that $d + 1$ should be less or equal to the number of eigenvalues of $M$ that are close to zero.

**Proposition 1.** Assume that the data $X_i \in \mathbb{R}^D$ is $K$-connected and that it is locally flat, i.e. there exists a corresponding set $Y_i \in \mathbb{R}^d$ for some $d > 0$ such that $Y_i = \sum_j W_{ij} Y_j$ (zero-error approximation), the set $\{Y_i\}$ has rank $d$, and has the origin as center of gravity: $\sum_{i=1}^N Y_i = 0$. Call $z$ the number of zero eigenvalues of the matrix $M$. Then $d < z$.

**Proof.** By construction the $N$ vector composed of all 1's is a zero-eigenvector of $M$. Moreover, since the $Y_i$ are such that the addends of $\Phi$ have zero error, then the matrix $Y$, which by hypothesis has rank $d$, is in the kernel of $I - W$ and hence in the kernel of $M$. Due to the center of gravity constraint, all the columns of $Y$ are orthogonal to the all 1's vector. Hence $M$ has at least $d + 1$ zero eigenvalues. $\square$

Therefore, in order to estimate $d$, one may count the number $z$ of zero eigenvalues of $M$ and choose any $d < z$. Within this range, smaller values of $d$ will yield more compact representations, while larger values of $d$ will yield more expressive ones, i.e. ones that are most faithful to the original data.

What happens in non-ideal conditions, i.e. when the data are not exactly locally flat, and when one has to contend with numerical noise? The appendix provides an argument showing that the statement in the proposition is robust with respect to

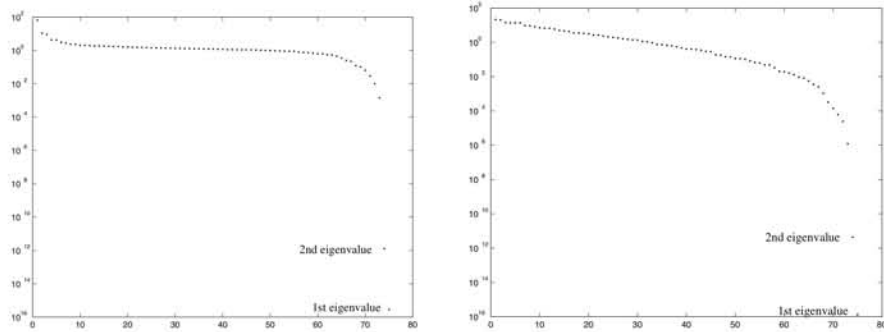

Figure 3: (Left) Eigenvalues for the straight-line data $X_i$ used for Figure 2. (Right) Eigenvalues for the curve data shown in the top-left panel of Figure 1. In both cases the two last eigenvalue are orders of magnitude smaller than the other eigenvalues, indicating a maximal dimension $d = 1$ for the data.

noise, i.e. numerical errors and small deviations from the ideal locally flat data will result in small deviations from the ideal zero-value of the first $d + 1$ eigenvalues, where $d$ is used here for the 'intrinsic' dimension of the data. This is illustrated in Figure 3.

In Figure 4 we describe the successful application of the dimensionality detection method on a data set of synthetically generated grayscale images.

## 4  LLE and grouping

In the first example (2.1) we pointed out the limits of LLE when applied to multiple components of data. It appears then that a grouping procedure should always preceed LLE. The data would be first split into its component groups, each one of which should be then analyzed with LLE. A deeper analysis of the algorithm though, suggests that grouping and LLE could actually be performed at the same time.

**Proposition 2.** Suppose the data set $\{X_i\}_{i=1,\dots,N} \in \mathbb{R}^D$ is partitioned into $m$ $K$-connected components. Then there exists an $m$-dimensional eigenspace of $M$ with zero eigenvalue which admits a basis $\{v_i\}_{i=1,\dots,m}$ where the $v_i$ have entries that are either '1' or '0'. More precisely: each $v_i$ corresponds to one of the groups of the data and takes value $v_{i,j} = 1$ for $j$ in the group, $v_{i,j} = 0$ for $j$ not in the group.

**Proof.** Without loss of generality, assume that the indexing of the data $X_i$ is such that the weight matrix $W$, and consequentely the matrix $M$, are block-diagonal with $m$ blocks, each block corresponding to one of the groups of data. This is achieved by a permutation of indices, which will not effect any further step of our algorithm. As a direct consequence of the row normalization of $W$, each block of $M$ has exactly one eigenvector composed of all ones, with eigenvalue 0. Therefore, there is an $m$-dimensional eigenspace with eigenvalue 0, and there exist a basis of it, each vector of which has value 1 on a certain component, 0 otherwise. $\square$

Therefore one may count the number of connected components by computing the eigenvectors of $M$ corresponding to eigenvalue 0, and counting the number $m$ of those vectors $v_i$ whose components take few discrete values (see Figure 6). Each index $i$ may be assigned to a group by clustering based on the value of $v_1, \dots, v_m$.

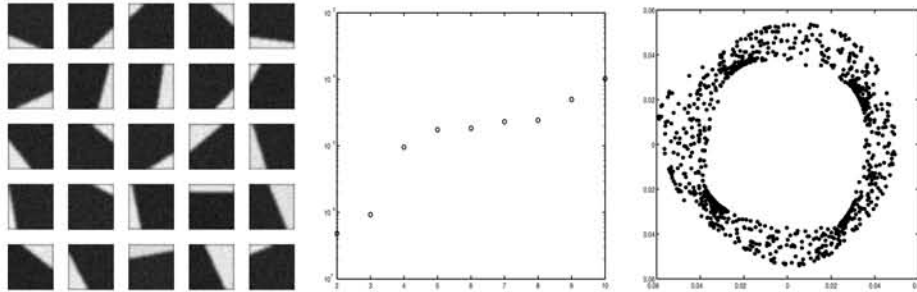

Figure 4: (Left) A sample from a data set of N=1000, 40 by 40 grayscale images, each one thought as a point in a 1600 dimensional vector space. In each image, a slightly blurred line separates a dark from a bright portion. The orientation of the line and its distance from the center of the image are variable. (Middle) The non-zero eigenvalues of M. LLE is performed with K=20. The 2nd and 3rd smallest eigenvalues are of smaller size than the others, giving an upper bound of 2 on the intrinsic dimension of the data set. (Right) The 2-dimensional LLE representation. The polar coordinates, after rescaling, are the distance of the dividing line from the center and its orientation.

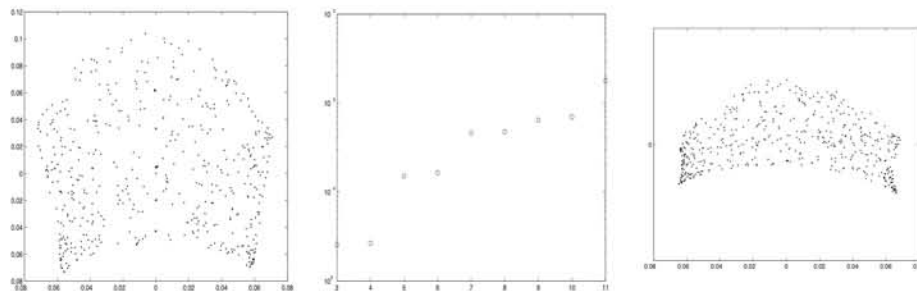

Figure 5: The data set is analogous to the one used above (N=1000, 40 by 40 grayscale images, LLE performed with K=20). The orientation of the line dividing the dark from the bright portion is now only allowed to vary in two disjoint intervals. (Middle) The non-zero eigenvalues of M. (Left and Right) The 3rd and 5th (resp. 4th and 6th) eigenvectors of M are used for the LLE representation of the first (resp. the second) K-component.

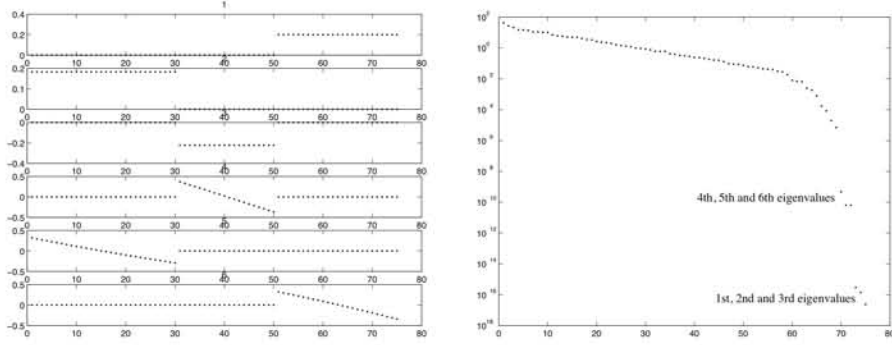

Figure 6: (Left) The last six eigenvectors of $M$ for the broken parabola of Figure 1 shown, top to bottom, in reverse order of magnitude of the corresponding eigenvalue. The $x$ axis is associated to the index $i$. (Right) The eigenvalues of the same (log scale). Notice that the last six are practically zero. The eigenvectors corresponding to the three last eigenvalues have discrete values indicating that the data is split in three groups. There are $z=6$ zero-eigenvalues indicating that the dimension of the data is $d \leq z/m - 1 = 1$.

In the Appendix (A) we show that such a process is robust with respect to numerical noise. It is also robust to small perturbations of the block-diagonal structure of $M$ (see Figure 7). This makes the use of LLE for grouping purposes convenient. Should the K-connected components be completely separated, the partition would be easily obtained via a more efficient graph-search algorithm.

The proof is carried out for ordered indices as in Fig. 3 but it is invariant under index permutation.

The analysis of Proposition 1 may be extended to the dimension of each of the $m$ groups according to Proposition 2. Therefore, in the ideal case, we will find $z$ zero-eigenvalues of $M$ which, together with the number $m$ obtained by counting the discrete-valued eigenvectors may be used to estimate the maximal $d$ using $z \geq m(d+1)$. This behavior may be observed experimentally, see Figures 6 and 5.

## 5   Conclusions

We have examined two difficulties of the Locally Linear Embedding method [2] and shown that, in a neighborhood of ideal conditions, they may be solved by a careful exam of eigenvectors of the matrix $M$ that are associated to very small eigenvalues.

More specifically: the number of groups in which the data is partitioned corresponds to the number of discrete-valued eigenvectors, while the maximal dimension $d$ of the low-dimensional embedding may be obtained by dividing the number of small eigenvalues by $m$ and subtracting 1.

Both the groups and the low-dimensional embedding coordinates may be computed from the components of such eigenvectors.

Our algorithms have mainly been tested on synthetically generated data. Further investigation on real data sets is necessary in order to validate our theoretical results.

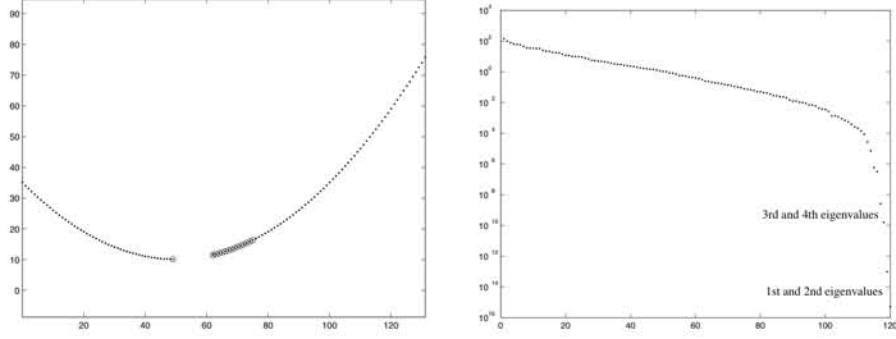

Figure 7: (Left) 2D Data $X_i$ distributed along a broken parabola. Nevertheless, for K=14, the components are not completely K-disconnected (a different symbol is used for the neighbors of the leftmost point on the rightmost component). (Right) The set of eigenvalues for M. A set of two almost-zero eigenvalues and a set of two of small size are visible.

# References

[1] *C. Bishop*, Neural Networks for Pattern Recognition, Oxford Univ. Press, (1995).

[2] *S.T. Roweis, L.K.Saul*, Science, **290**, p. 2323-2326, (2000).

[3] *J. Tenenbaum, V. de Silva, J. Langford*, Science, **290**, p. 2319-2323, (2000).

# A  Appendix

In Proposition 2 of Section 4 we proved that during the LLE procedure we can automatically detect the number of $K$-connected components, in case there is no noise. Similarly, in Proposition 1 of Section 3 we proved that under ideal conditions (no noise, locally flat data), we can determine an estimate for the intrinsic dimension of the data. Our next goal is to establish a certain robustness of these results in the case there is numerical noise, or the components are not completely separated, or the data is not exactly locally flat.

In general, suppose we have a non degenerate matrix $A$, and an orthonormal basis of eigenvectors $v_1, ..., v_m$, with eigenvalues $\lambda_1, ...\lambda_m$. As a consequence of a small perturbation of the matrix into $A + dA$, we will have eigenvectors $v_i + dv_i$ with eigenvalues $\lambda_i + d\lambda_i$. The unitary norm constraint makes sure that $dv_i$ is orthogonal to $v_i$ and could be therefore written as $dv_i = \sum_{k \neq i} \alpha_{ik} v_k$. Using again the orthonormality, one can derive expressions for the perturbations of $\lambda_i$ and $v_i$:

$$
\begin{aligned}
d\lambda_i &= \ <v_i, dAv_i> \\
\alpha_{ij}(\lambda_i - \lambda_j) &= \ <v_j, dAv_i> .
\end{aligned}
$$

This shows that if the perturbation $dA$ has order $\epsilon$, then the perturbations $d\lambda$ and $\alpha_{ij}$ are also of order $\epsilon$. Notice that we are not interested in perturbations $\alpha_{ij}$ *within* the eigenspace of eigenvalue 0, but rather those *orthogonal* to it, and therefore $\lambda_i \neq \lambda_j$.